# The Multidimensional Wisdom of Crowds

**Peter Welinder**[1]   **Steve Branson**[2]   **Serge Belongie**[2]   **Pietro Perona**[1]
[1] California Institute of Technology, [2] University of California, San Diego
{welinder,perona}@caltech.edu   {sbranson,sjb}@cs.ucsd.edu

## Abstract

Distributing labeling tasks among hundreds or thousands of annotators is an increasingly important method for annotating large datasets. We present a method for estimating the underlying value (e.g. the class) of each image from (noisy) annotations provided by multiple annotators. Our method is based on a model of the image formation and annotation process. Each image has different characteristics that are represented in an abstract Euclidean space. Each annotator is modeled as a multidimensional entity with variables representing competence, expertise and bias. This allows the model to discover and represent groups of annotators that have different sets of skills and knowledge, as well as groups of images that differ qualitatively. We find that our model predicts ground truth labels on both synthetic and real data more accurately than state of the art methods. Experiments also show that our model, starting from a set of binary labels, may discover rich information, such as different "schools of thought" amongst the annotators, and can group together images belonging to separate categories.

## 1   Introduction

Producing large-scale training, validation and test sets is vital for many applications. Most often this job has to be carried out "by hand" and thus it is delicate, expensive, and tedious. Services such as Amazon Mechanical Turk (MTurk) have made it easy to distribute simple labeling tasks to hundreds of workers. Such "crowdsourcing" is increasingly popular and has been used to annotate large datasets in, for example, Computer Vision [8] and Natural Language Processing [7]. As some annotators are unreliable, the common wisdom is to collect multiple labels per exemplar and rely on "majority voting" to determine the correct label. We propose a model for the annotation process with the goal of obtaining more reliable labels with as few annotators as possible.

It has been observed that some annotators are more skilled and consistent in their labels than others. We postulate that the ability of annotators is multidimensional; that is, an annotator may be good at some aspects of a task but worse at others. Annotators may also attach different costs to different kinds of errors, resulting in different biases for the annotations. Furthermore, different pieces of data may be easier or more difficult to label. All of these factors contribute to a "noisy" annotation process resulting in inconsistent labels. Although approaches for modeling certain aspects of the annotation process have been proposed in the past [1, 5, 6, 9, 13, 4, 12], no attempt has been made to blend all characteristics of the process into a single unified model.

This paper has two main contributions: (1) we improve on current state-of-the-art methods for crowdsourcing by introducing a more comprehensive and accurate model of the human annotation process, and (2) we provide insight into the human annotation process by learning a richer representation that distinguishes amongst the different sources of annotator error. Understanding the annotation process can be important toward quantifying the extent to which datasets constructed from human data are "ground truth".

We propose a generative Bayesian model for the annotation process. We describe an inference algorithm to estimate the properties of the data being labeled and the annotators labeling them. We show on synthetic and real data that the model can be used to estimate data difficulty and annotator

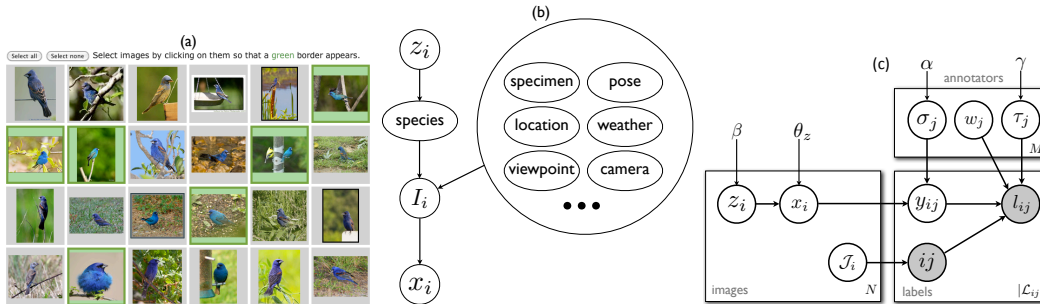

Figure 1: (a) Sample MTurk task where annotators were asked to click on images of Indigo Bunting (described in Section 5.2). (b) The image formation process. The class variable $z_i$ models if the object (Indigo Bunting) will be present ($z_i = 1$) or absent ($z_i = 0$) in the image, while a number of "nuisance factors" influence the appearance of the image. The image is then transformed into a low-dimensional representation $x_i$ which captures the main attributes that are considered by annotators in labeling the image. (c) Probabilistic graphical model of the entire annotation process where image formation is summarized by the nodes $z_i$ and $x_i$. The observed variables, indicated by shaded circles, are the index $i$ of the image, index $j$ of the annotators, and value $l_{ij}$ of the label provided by annotator $j$ for image $i$. The annotation process is repeated for all $i$ and for multiple $j$ thus obtaining multiple labels per image with each annotator labeling multiple images (see Section 3).

biases, while identifying annotators' different "areas of strength". While many of our results are valid for general labels and tasks, we focus on the binary labeling of images.

## 2   Related Work

The advantages and drawbacks of using crowdsourcing services for labeling large datasets have been explored by various authors [2, 7, 8]. In general, it has been found that many labels are of high quality [8], but a few sloppy annotators do low quality work [7, 12]; thus the need for efficient algorithms for integrating the labels from many annotators [5, 12]. A related topic is that of using paired games for obtaining annotations, which can be seen as a form of crowdsourcing [10, 11].

Methods for combining the labels from many different annotators have been studied before. Dawid and Skene [1] presented a model for multi-valued annotations where the biases and skills of the annotators were modeled by a confusion matrix. This model was generalized and extended to other annotation types by Welinder and Perona [12]. Similarly, the model presented by Raykar et al. [4] considered annotator bias in the context of training binary classifiers with noisy labels. Building on these works, our model goes a step further in modeling each annotator as a multidimensional classifier in an abstract feature space. We also draw inspiration from Whitehill et al. [13], who modeled both annotator competence and image difficulty, but did not consider annotator bias. Our model generalizes [13] by introducing a high-dimensional concept of image difficulty and combining it with a broader definition of annotator competence. Other approaches have been proposed for non-binary annotations [9, 6, 12]. By modeling annotator competence and image difficulty as multidimensional quantities, our approach achieves better performance on real data than previous methods and provides a richer output space for separating groups of annotators and images.

## 3   The Annotation Process

An annotator, indexed by $j$, looks at image $I_i$ and assigns it a label $l_{ij}$. *Competent* annotators provide accurate and precise labels, while unskilled annotators provide inconsistent labels. There is also the possibility of adversarial annotators assigning labels that are opposite to those assigned by competent annotators. Annotators may have different areas of strength, or *expertise*, and thus provide more reliable labels on different subsets of images. For example, when asked to label images containing ducks some annotators may be more aware of the distinction between ducks and geese while others may be more aware of the distinction between ducks, grebes, and cormorants (visually similar bird species). Furthermore, different annotators may weigh errors differently; one annotator may be intolerant of false positives, while another is more *optimistic* and accepts the cost of a few false positives in order to get a higher detection rate. Lastly, the *difficulty* of the image may also matter. A difficult or ambiguous image may be labeled inconsistently even by competent annotators, while an easy image is labeled consistently even by sloppy annotators. In modeling the annotation process, all of these factors should be considered.

We model the annotation process in a sequence of steps. $N$ images are produced by some image capture/collection process. First, a variable $z_i$ decides which set of "objects" contribute to producing an image $I_i$. For example, $z_i \in \{0, 1\}$ may denote the presence/absence of a particular bird species. A number of "nuisance factors," such as viewpoint and pose, determine the image (see Figure 1).

Each image is transformed by a deterministic "visual transformation" converting pixels into a vector of task-specific measurements $x_i$, representing measurements that are available to the visual system of an ideal annotator. For example, the $x_i$ could be the firing rates of task-relevant neurons in the brain of the best human annotator. Another way to think about $x_i$ is that it is a vector of visual attributes (beak shape, plumage color, tail length etc) that the annotator will consider when deciding on a label. The process of transforming $z_i$ to the "signal" $x_i$ is stochastic and it is parameterized by $\theta_z$, which accounts for the variability in image formation due to the nuisance factors.

There are $M$ annotators in total, and the set of annotators that label image $i$ is denoted by $\mathcal{J}_i$. An annotator $j \in \mathcal{J}_i$, selected to label image $I_i$, does not have direct access to $x_i$, but rather to $y_{ij} = x_i + n_{ij}$, a version of the signal corrupted by annotator-specific and image-specific "noise" $n_{ij}$. The noise process models differences between the measurements that are ultimately available to individual annotators. These differences may be due to visual acuity, attention, direction of gaze, etc. The statistics of this noise are different from annotator to annotator and are parametrized by $\sigma_j$. Most significantly, the variance of the noise will be lower for competent annotators, as they are more likely to have access to a clearer and more consistent representation of the image than confused or unskilled annotators.

The vector $y_{ij}$ can be understood as a perceptual encoding that encompasses all major components that affect an annotator's judgment on an annotation task. Each annotator is parameterized by a unit vector $\hat{w}_j$, which models the annotator's individual weighting on each of these components. In this way, $\hat{w}_j$ encodes the training or expertise of the annotator in a multidimensional space. The scalar projection $\langle y_{ij}, \hat{w}_j \rangle$ is compared to a threshold $\hat{\tau}_j$. If the signal is above the threshold, the annotator assigns a label $l_{ij} = 1$, and $l_{ij} = 0$ otherwise.

## 4 Model and Inference

Putting together the assumptions of the previous section, we obtain the graphical model shown in Figure 1. We will assume a Bayesian treatment, with priors on all parameters. The joint probability distribution, excluding hyper-parameters for brevity, can be written as

$$p(\mathcal{L}, z, x, y, \sigma, \hat{w}, \hat{\tau}) = \prod_{j=1}^{M} p(\sigma_j) p(\hat{\tau}_j) p(\hat{w}_j) \prod_{i=1}^{N} \left( p(z_i) p(x_i \mid z_i) \prod_{j \in \mathcal{J}_i} p(y_{ij} \mid x_i, \sigma_j) \, p(l_{ij} \mid \hat{w}_j, \hat{\tau}_j, y_{ij}) \right),$$
(1)

where we denote $z$, $x$, $y$, $\sigma$, $\hat{\tau}$, $\hat{w}$, and $\mathcal{L}$ to mean the sets of all the corresponding subscripted variables. This section describes further assumptions on the probability distributions. These assumptions are not necessary; however, in practice they simplify inference without compromising the quality of the parameter estimates.

Although both $z_i$ and $l_{ij}$ may be continuous or multivalued discrete in a more general treatment of the model [12], we henceforth assume that they are binary, i.e. $z_i, l_{ij} \in \{0, 1\}$. We assume a Bernoulli prior on $z_i$ with $p(z_i = 1) = \beta$, and that $x_i$ is normally distributed[1] with variance $\theta_z^2$,

$$p(x_i \mid z_i) = \mathcal{N}(x_i; \, \mu_z, \theta_z^2),$$
(2)

where $\mu_z = -1$ if $z_i = 0$ and $\mu_z = 1$ if $z_i = 1$ (see Figure 2a). If $x_i$ and $y_{ij}$ are multi-dimensional, then $\sigma_j$ is a covariance matrix. These assumptions are equivalent to using a mixture of Gaussians prior on $x_i$.

The noisy version of the signal $x_i$ that annotator $j$ sees, denoted by $y_{ij}$, is assumed to be generated by a Gaussian with variance $\sigma_j^2$ centered at $x_i$, that is $p(y_{ij} \mid x_i, \sigma_j) = \mathcal{N}(y_{ij}; \, x_i, \sigma_j^2)$ (see Figure 2b). We assume that each annotator assigns the label $l_{ij}$ according to a linear classifier. The classifier is parameterized by a direction $\hat{w}_j$ of a decision plane and a bias $\hat{\tau}_j$. The label $l_{ij}$ is deterministically chosen, i.e. $l_{ij} = \mathbb{I}(\langle \hat{w}_j, y_{ij} \rangle \geq \hat{\tau}_j)$, where $\mathbb{I}(\cdot)$ is the indicator function. It is possible to integrate

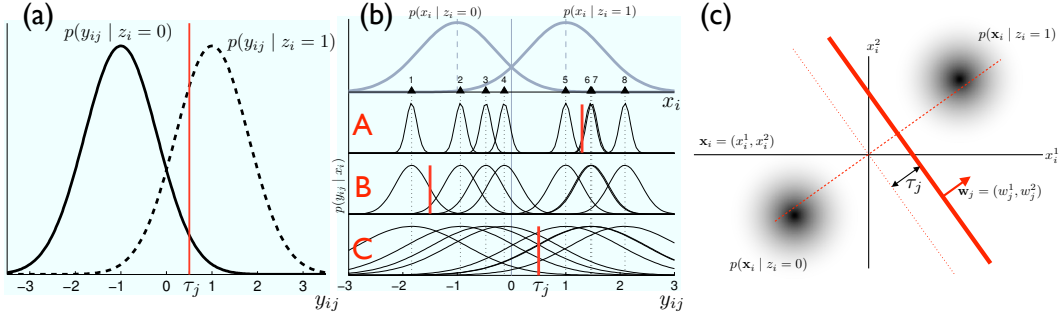

Figure 2: Assumptions of the model. (a) Labeling is modeled in a signal detection theory framework, where the signal $y_{ij}$ that annotator $j$ sees for image $I_i$ is produced by one of two Gaussian distributions. Depending on $y_{ij}$ and annotator parameters $w_j$ and $\tau_j$, the annotator labels 1 or 0. (b) The image representation $x_i$ is assumed to be generated by a Gaussian mixture model where $z_i$ selects the component. The figure shows 8 different realizations $x_i$ ($x_1, \ldots, x_8$), generated from the mixture model. Depending on the annotator $j$, noise $n_{ij}$ is added to $x_i$. The three lower plots shows the noise distributions for three different annotators (A,B,C), with increasing "incompetence" $\sigma_j$. The biases $\tau_j$ of the annotators are shown with the red bars. Image no. 4, represented by $x_4$, is the most ambiguous image, as it is very close to the optimal decision plane at $x_i = 0$. (c) An example of 2-dimensional $x_i$. The red line shows the decision plane for one annotator.

out $y_{ij}$ and put $l_{ij}$ in direct dependence on $x_i$,

$$p(l_{ij} = 1 \mid x_i, \sigma_j, \hat{\tau}_j) = \Phi\left(\frac{\langle \hat{w}_j, x_i \rangle - \hat{\tau}_j}{\sigma_j}\right), \tag{3}$$

where $\Phi(\cdot)$ is the cumulative standardized normal distribution, a sigmoidal-shaped function.

In order to remove the constraint on $\hat{w}_j$ being a direction, i.e. $\|\hat{w}_j\|^2 = 1$, we reparameterize the problem with $w_j = \hat{w}_j/\sigma_j$ and $\tau_j = \hat{\tau}_j/\sigma_j$. Furthermore, to regularize $w_j$ and $\tau_j$ during inference, we give them Gaussian priors parameterized by $\alpha$ and $\gamma$ respectively. The prior on $\tau_j$ is centered at the origin and is very broad ($\gamma = 3$). For the prior on $w_j$, we kept the center close to the origin to be initially pessimistic of the annotator competence, and to allow for adversarial annotators (mean 1, std 3). All of the hyperparameters were chosen somewhat arbitrarily to define a scale for the parameter space, and in our experiments we found that results (such as error rates in Figure 3) were quite insensitive to variations in the hyperparameters. The modified Equation 1 becomes,

$$p(\mathcal{L}, x, w, \tau) = \prod_{j=1}^{M} p(\tau_j \mid \gamma) p(w_j \mid \alpha) \prod_{i=1}^{N} \left( p(x_i \mid \theta_z, \beta) \prod_{j \in \mathcal{J}_i} p(l_{ij} \mid x_i, w_j, \tau_j) \right). \tag{4}$$

The only observed variables in the model are the labels $\mathcal{L} = \{l_{ij}\}$, from which the other parameters have to be inferred. Since we have priors on the parameters, we proceed by MAP estimation, where we find the optimal parameters $(x^\star, w^\star, \tau^\star)$ by maximizing the posterior on the parameters,

$$(x^\star, w^\star, \tau^\star) = \arg\max_{x,w,\tau} p(x, w, \tau \mid \mathcal{L}) = \arg\max_{x,w,\tau} m(x, w, \tau), \tag{5}$$

where we have defined $m(x, w, \tau) = \log p(\mathcal{L}, x, w, \tau)$ from Equation 4. Thus, to do inference, we need to optimize

$$m(x, w, \tau) = \sum_{i=1}^{N} \log p(x_i \mid \theta_z, \beta) + \sum_{j=1}^{M} \log p(w_j \mid \alpha) + \sum_{j=1}^{M} \log p(\tau_j \mid \gamma)$$

$$+ \sum_{i=1}^{N} \sum_{j \in \mathcal{J}_i} \left[ l_{ij} \log \Phi\left(\langle w_j, x_i \rangle - \tau_j\right) + (1 - l_{ij}) \log\left(1 - \Phi\left(\langle w_j, x_i \rangle - \tau_j\right)\right) \right]. \tag{6}$$

To maximize (6) we carry out alternating optimization using gradient ascent. We begin by fixing the $x$ parameters and optimizing Equation 6 for $(w, \tau)$ using gradient ascent. Then we fix $(w, \tau)$ and optimize for $x$ using gradient ascent, iterating between fixing the image parameters and annotator parameters back and forth. Empirically, we have observed that this optimization scheme usually converges within 20 iterations.

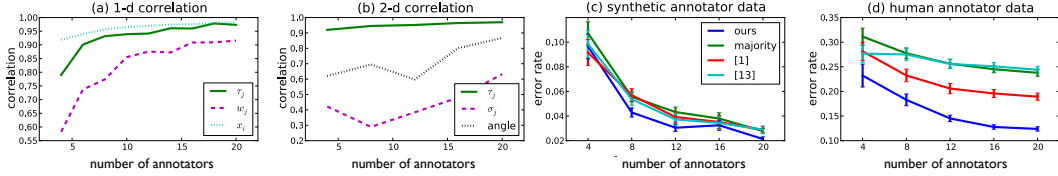

Figure 3: (a) and (b) show the correlation between the ground truth and estimated parameters as the number of annotators increases on synthetic data for 1-d and 2-d $x_i$ and $w_j$. (c) Performance of our model in predicting $z_i$ on the data from (a), compared to majority voting, the model of [1], and GLAD [13]. (d) Performance on real labels collected from MTurk. See Section 5.1 for details on (a-c) and Section 5.2 for details on (d).

In the derivation of the model above, there is no restriction on the dimensionality of $x_i$ and $w_j$; they may be one-dimensional scalars or higher-dimensional vectors. In the former case, assuming $\hat{w}_j = 1$, the model is equivalent to a standard signal detection theoretic model [3] where a signal $y_{ij}$ is generated by one of two Normal distributions $p(y_{ij} \mid z_i) = \mathcal{N}(y_{ij} \mid \mu_z, s^2)$ with variance $s^2 = \theta_z^2 + \sigma_j^2$, centered on $\mu_0 = -1$ and $\mu_1 = 1$ for $z_i = 0$ and $z_i = 1$ respectively (see Figure 2a). In signal detection theory, the sensitivity index, conventionally denoted $d'$, is a measure of how well the annotator can discriminate the two values of $z_i$ [14]. It is defined as the Mahalanobis distance between $\mu_0$ and $\mu_1$ normalized by $s$,

$$d' = \frac{\mu_1 - \mu_0}{s} = \frac{2}{\sqrt{\theta_z^2 + \sigma_j^2}}. \tag{7}$$

Thus, the lower $\sigma_j$, the better the annotator can distinguish between classes of $z_i$, and the more "competent" he is. The sensitivity index can also be computed directly from the false alarm rate $f$ and hit rate $h$ using $d' = \Phi^{-1}(h) - \Phi^{-1}(f)$ where $\Phi^{-1}(\cdot)$ is the inverse of the cumulative normal distribution [14]. Similarly, the "threshold", which is a measure of annotator bias, can be computed by $\lambda = -\frac{1}{2}\left(\Phi^{-1}(h) + \Phi^{-1}(f)\right)$. A large positive $\lambda$ means that the annotator attributes a high cost to false positives, while a large negative $\lambda$ means the annotator avoids false negative mistakes. Under the assumptions of our model, $\lambda$ is related to $\tau_j$ in our model by the relation $\lambda = \hat{\tau}_j/s$.

In the case of higher dimensional $x_i$ and $w_j$, each component of the $x_i$ vector can be thought of as an attribute or a high level feature. For example, the task may be to label only images with a particular bird species, say "duck", with label 1, and all other images with 0. Some images contain no birds at all, while other images contain birds similar to ducks, such as geese or grebes. Some annotators may be more aware of the distinction between ducks and geese and others may be more aware of the distinction between ducks, grebes and cormorants. In this case, $x_i$ can be considered to be 2-dimensional. One dimension represents image attributes that are useful in the distinction between ducks and geese, and the other dimension models parameters that are useful in distinction between ducks and grebes (see Figure 2c). Presumably all annotators see the same attributes, signified by $x_i$, but they use them differently. The model can distinguish between annotators with preferences for different attributes, as shown in Section 5.2.

Image difficulty is represented in the model by the value of $x_i$ (see Figure 2b). If there is a particular ground truth decision plane, $(w', \tau')$, images $I_i$ with $x_i$ close to the plane will be more difficult for annotators to label. This is because the annotators see a noise corrupted version, $y_{ij}$, of $x_i$. How well the annotators can label a particular image depends on both the closeness of $x_i$ to the ground truth decision plane and the annotator's "noise" level, $\sigma_j$. Of course, if the annotator bias $\tau_j$ is far from the ground truth decision plane, the labels for images near the ground truth decision plane will be consistent for that annotator, but not necessarily correct.

# 5 Experiments

## 5.1 Synthetic Data

To explore whether the inference procedure estimates image and annotator parameters accurately, we tested our model on synthetic data generated according to the model's assumptions. Similar to the experimental setup in [13], we generated 500 synthetic image parameters and simulated between 4 and 20 annotators labeling each image. The procedure was repeated 40 times to reduce the noise in the results.

We generated the annotator parameters by randomly sampling $\sigma_j$ from a Gamma distribution (shape 1.5 and scale 0.3) and biases $\tau_j$ from a Normal distribution centered at 0 with standard deviation

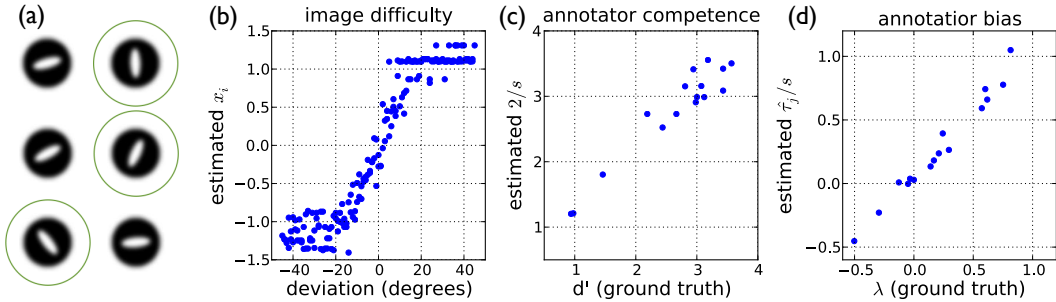

Figure 4: Ellipse dataset. (a) The images to be labeled were fuzzy ellipses (oriented uniformly from 0 to $\pi$) enclosed in dark circles. The task was to select ellipses that were more vertical than horizontal (the former are marked with green circles in the figure). (b-d) The image difficulty parameters $x_i$, annotator competence $2/s$, and bias $\hat{\tau}_j/s$ learned by our model are compared to the ground truth equivalents. The closer $x_i$ is to 0, the more ambiguous/difficult the discrimination task, corresponding to ellipses that have close to $45°$ orientation.

0.5. The direction of the decision plane $w_j$ was $+1$ with probability 0.99 and $-1$ with probability 0.01. The image parameters $x_i$ were generated by a two-dimensional Gaussian mixture model with two components of standard deviation 0.8 centered at -1 and +1. The image ground truth label $z_i$, and thus the mixture component from which $x_i$ was generated, was sampled from a Bernoulli distribution with $p(z_i = 1) = 0.5$.

For each trial, we measured the correlation between the ground truth values of each parameter and the values estimated by the model. We averaged Spearman's rank correlation coefficient for each parameter over all trials. The result of the simulated labeling process is shown Figure 3a. As can be seen from the figure, the model estimates the parameters accurately, with the accuracy increasing as the number of annotators labeling each image increases. We repeated a similar experiment with 2-dimensional $x_i$ and $w_j$ (see Figure 3b). As one would expect, estimating higher dimensional $x_i$ and $w_j$ requires more data.

We also examined how well our model estimated the binary class values, $z_i$. For comparison, we also tried three other methods on the same data: a simple majority voting rule for each image, the bias-competence model of [1], and the GLAD algorithm from [13][2], which models 1-d image difficulty and annotator competence, but not bias. As can be seen from Figure 3c, our method presents a small but consistent improvement. In a separate experiment (not shown) we generated synthetic annotators with increasing bias parameters $\tau_j$. We found that GLAD performs worse than majority voting when the variance in the bias between different annotators is high ($\gamma \gtrsim 0.8$); this was expected as GLAD does not model annotator bias. Similarly, increasing the proportion of difficult images degrades the performance of the model from [1]. The performance of our model points to the benefits of modeling all aspects of the annotation process.

## 5.2 Human Data

We next conducted experiments on annotation results from real MTurk annotators. To compare the performance of the different models on a real discrimination task, we prepared dataset of 200 images of birds (100 with Indigo Bunting, and 100 with Blue Grosbeak), and asked 40 annotators per image if it contained at least one Indigo Bunting; this is a challenging task (see Figure 1). The annotators were given a description and example photos of the two bird species. Figure 3d shows how the performance varies as the number of annotators per image is increased. We sampled a subset of the annotators for each image. Our model did better than the other approaches also on this dataset.

To demonstrate that annotator competence, annotator bias, image difficulty, and multi-dimensional decision surfaces are important real life phenomena affecting the annotation process, and to quantify our model's ability to adapt to each of them, we tested our model on three different image datasets: one based on pictures of rotated ellipses, another based on synthetically generated "greebles", and a third dataset with images of waterbirds.

**Ellipse Dataset:** Annotators were given the simple task of selecting ellipses which they believed to be more vertical than horizontal. This dataset was chosen to make the model's predictions quan-

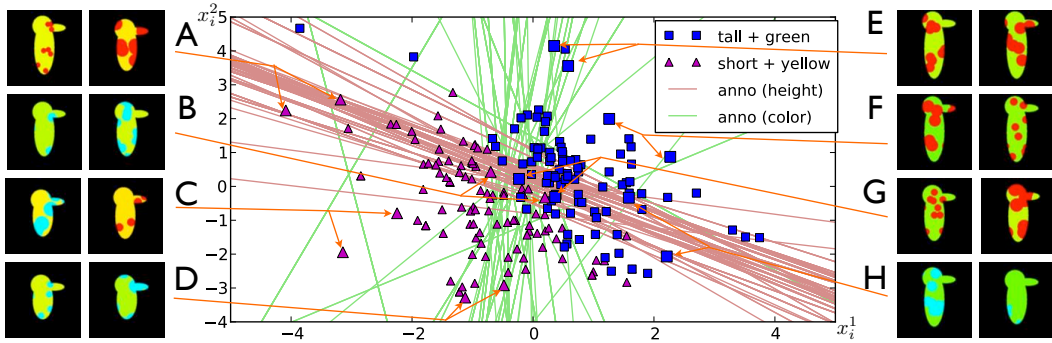

Figure 5: Estimated image parameters (symbols) and annotator decision planes (lines) for the greeble experiment. Our model learns two image parameter dimensions $x_i^1$ and $x_i^2$ which roughly correspond to color and height, and identifies two clusters of annotator decision planes, which correctly correspond to annotators primed with color information (green lines) and height information (red lines). On the left are example images of class 1, which are shorter and more yellow (red and blue dots are uncorrelated with class), and on the right are images of class 2, which are taller and more green. $C$ and $F$ are easy for all annotators, $A$ and $H$ are difficult for annotators that prefer height but easy for annotators that prefer color, $D$ and $E$ are difficult for annotators that prefer color but easy for annotators that prefer height, $B$ and $G$ are difficult for all annotators.

tifiable, because ground truth class labels and ellipse angle parameters are known to us for each test image (but hidden from the inference algorithm).

By definition, ellipses at an angle of $45°$ are impossible to classify, and we expect that images gradually become easier to classify as the angle moves away from $45°$. We used a total of 180 ellipse images, with rotation angle varying from 1-$180°$, and collected labels from 20 MTurk annotators for each image. In this dataset, the estimated image parameters $x_i$ and annotator parameters $w_j$ are 1-dimensional, where the magnitudes encode image difficulty and annotator competence respectively. Since we had ground truth labels, we could compute the false alarm and hit rates for each annotator, and thus compute $\lambda$ and $d'$ for comparison with $\hat{\tau}_j/s$ and $2/s$ (see Equation 7 and following text).

The results in Figure 4b-d show that annotator competence and bias vary among annotators. Moreover, the figure shows that our model accurately estimates image difficulty, annotator competence, and annotator bias on data from real MTurk annotators.

**Greeble Dataset:** In the second experiment, annotators were shown pictures of "greebles" (see Figure 5) and were told that the greebles belonged to one of two classes. Some annotators were told that the two greeble classes could be discriminated by height, while others were told they could be discriminated by color (yellowish vs. green). This was done to explore the scenario in which annotators have different types of prior knowledge or abilities. We used a total of 200 images with 20 annotators labeling each image. The height and color parameters for the two types of greebles were randomly generated according to Gaussian distributions with centers $(1, 1)$ and $(-1, -1)$, and standard deviations of $0.8$.

The results in Figure 5 show that the model successfully learned two clusters of annotator decision surfaces, one (green) of which responds mostly to the first dimension of $x_i$ (color) and another (red) responding mostly to the second dimension of $x_i$ (height). These two clusters coincide with the sets of annotators primed with the two different attributes. Additionally, for the second attribute, we observed a few "adversarial" annotators whose labels tended to be inverted from their true values. This was because the instructions to our color annotation task were ambiguously worded, so that some annotators had become confused and had inverted their labels. Our model robustly handles these adversarial labels by inverting the sign of the $\hat{w}$ vector.

**Waterbird Dataset**: The greeble experiment shows that our model is able to segregate annotators looking for different attributes in images. To see whether the same phenomenon could be observed in a task involving images of real objects, we constructed an image dataset of waterbirds. We collected 50 photographs each of the bird species Mallard, American Black Duck, Canada Goose and Red-necked Grebe. In addition to the 200 images of waterbirds, we also selected 40 images without any birds at all (such as photos of various nature scenes and objects) or where birds were too small be seen clearly, making 240 images in total. For each image, we asked 40 annotators on MTurk if they could see a duck in the image (only Mallards and American Black Ducks are ducks). The hypothesis

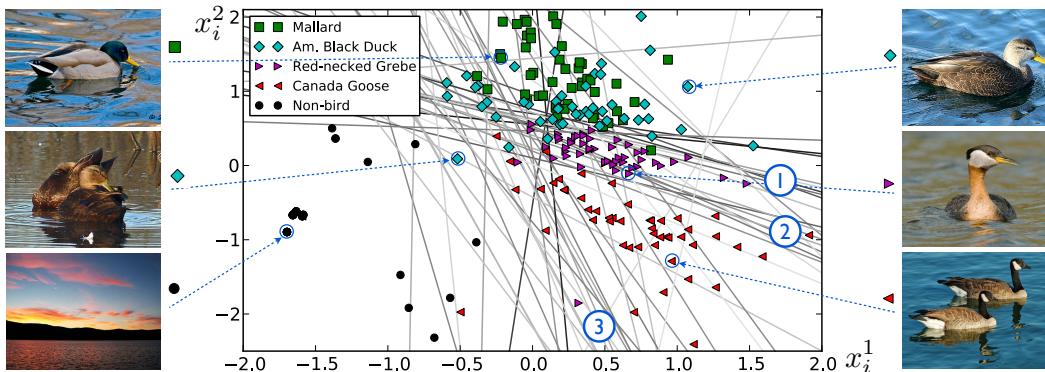

Figure 6: Estimated image and annotator parameters on the Waterbirds dataset. The annotators were asked to select images containing at least one "duck". The estimated $x_i$ parameters for each image are marked with symbols that are specific to the class the image belongs to. The arrows show the $x_i$ coordinates of some example images. The gray lines are the decision planes of the annotators. The darkness of the lines is an indicator of $\|w_j\|$: darker gray means the model estimated the annotator to be more competent. Notice how the annotators' decision planes fall roughly into three clusters, marked by the blue circles and discussed in Section 5.2.

was that some annotators would be able to discriminate ducks from the two other bird species, while others would confuse ducks with geese and/or grebes.

Results from the experiment, shown in Figure 6, suggest that there are at least three different groups of annotators, those who separate: (1) ducks from everything else, (2) ducks and grebes from everything else, and (3) ducks, grebes, and geese from everything else; see numbered circles in Figure 6. Interestingly, the first group of annotators was better at separating out Canada geese than Red-necked grebes. This may be because Canada geese are quite distinctive with their long, black necks, while the grebes have shorter necks and look more duck-like in most poses. There were also a few outlier annotators that did not provide answers consistent with any other annotators. This is a common phenomenon on MTurk, where a small percentage of the annotators will provide bad quality labels in the hope of still getting paid [7]. We also compared the labels predicted by the different models to the ground truth. Majority voting performed at 68.3% correct labels, GLAD at 60.4%, and our model performed at 75.4%.

## 6  Conclusions

We have proposed a Bayesian generative probabilistic model for the annotation process. Given only binary labels of images from many different annotators, it is possible to infer not only the underlying class (or value) of the image, but also parameters such as image difficulty and annotator competence and bias. Furthermore, the model represents both the images and the annotators as multidimensional entities, with different high level attributes and strengths respectively. Experiments with images annotated by MTurk workers show that indeed different annotators have variable competence level and widely different biases, and that the annotators' classification criterion is best modeled in multidimensional space. Ultimately, our model can accurately estimate the ground truth labels by integrating the labels provided by several annotators with different skills, and it does so better than the current state of the art methods.

Besides estimating ground truth classes from binary labels, our model provides information that is valuable for defining loss functions and for training classifiers. For example, the image parameters estimated by our model could be taken into account for weighing different training examples, or, more generally, it could be used for a softer definition of ground truth. Furthermore, our findings suggest that annotators fall into different groups depending on their expertise and on how they perceive the task. This could be used to select annotators that are experts on certain tasks and to discover different schools of thought on how to carry out a given task.

### Acknowledgements

P.P. and P.W. were supported by ONR MURI Grant #N00014-06-1-0734 and EVOLUT.ONR2. S.B. was supported by NSF CAREER Grant #0448615, NSF Grant AGS-0941760, ONR MURI Grant #N00014-08-1-0638, and a Google Research Award.

## Footnotes

[1]We used the parameters $\beta = 0.5$ and $\theta_z = 0.8$.

[2]We used the implementation of GLAD available on the first author's website: `http://mplab.ucsd.edu/~jake/` We varied the $\alpha$ prior in their code between 1-10 to achieve best performance.

# References

[1] A. P. Dawid and A. M. Skene. Maximum likelihood estimation of observer error-rates using the em algorithm. *J. Roy. Statistical Society, Series C*, 28(1):20–28, 1979. 1, 2, 5, 6

[2] J. Deng, W. Dong, R. Socher, L.-J. Li, K. Li, and L. Fei-Fei. ImageNet: A Large-Scale Hierarchical Image Database. In *CVPR*, 2009. 2

[3] D.M. Green and J.M. Swets. *Signal detection theory and psychophysics*. John Wiley and Sons Inc, New York, 1966. 5

[4] V.C. Raykar, S. Yu, L.H. Zhao, A. Jerebko, C. Florin, G.H. Valadez, L. Bogoni, and L. Moy. Supervised Learning from Multiple Experts: Whom to trust when everyone lies a bit. In *ICML*, 2009. 1, 2

[5] V.S. Sheng, F. Provost, and P.G. Ipeirotis. Get another label? improving data quality and data mining using multiple, noisy labelers. In *KDD*, 2008. 1, 2

[6] P. Smyth, U. Fayyad, M. Burl, P. Perona, and P. Baldi. Inferring ground truth from subjective labelling of Venus images. *NIPS*, 1995. 1, 2

[7] R. Snow, B. O'Connor, D. Jurafsky, and A.Y. Ng. Cheap and Fast - But is it Good? Evaluating Non-Expert Annotations for Natural Language Tasks. In *EMNLP*, 2008. 1, 2, 8

[8] A. Sorokin and D. Forsyth. Utility data annotation with amazon mechanical turk. In *First IEEE Workshop on Internet Vision at CVPR'08*, 2008. 1, 2

[9] M. Spain and P. Perona. Some objects are more equal than others: measuring and predicting importance. In *ECCV*, 2008. 1, 2

[10] L. von Ahn and L. Dabbish. Labeling images with a computer game. In *SIGCHI conference on Human factors in computing systems*, pages 319–326, 2004. 2

[11] L. von Ahn, B. Maurer, C. McMillen, D. Abraham, and M. Blum. reCAPTCHA: Human-based character recognition via web security measures. *Science*, 321(5895):1465–1468, 2008. 2

[12] Peter Welinder and Pietro Perona. Online crowdsourcing: rating annotators and obtaining cost-effective labels. In *IEEE Conference on Computer Vision and Pattern Recognition Workshops (ACVHL)*, 2010. 1, 2, 3

[13] J. Whitehill, P. Ruvolo, T. Wu, J. Bergsma, and J. Movellan. Whose vote should count more: Optimal integration of labels from labelers of unknown expertise. In *NIPS*, 2009. 1, 2, 5, 6

[14] T. D. Wickens. *Elementary signal detection theory*. Oxford University Press, United States, 2002. 5

